# SPERT-II: A Vector Microprocessor System and its Application to Large Problems in Backpropagation Training

**John Wawrzynek, Krste Asanović, & Brian Kingsbury**
University of California at Berkeley
Department of Electrical Engineering and Computer Sciences
Berkeley, CA 94720-1776
{johnw,krste,bedk}@cs.berkeley.edu

**James Beck, David Johnson, & Nelson Morgan**
International Computer Science Institute
1947 Center Street, Suite 600
Berkeley, CA 94704-1105
{beck,davidj,morgan}@icsi.berkeley.edu

## Abstract

We report on our development of a high-performance system for neural network and other signal processing applications. We have designed and implemented a vector microprocessor and packaged it as an attached processor for a conventional workstation. We present performance comparisons with commercial workstations on neural network backpropagation training. The SPERT-II system demonstrates significant speedups over extensively hand-optimization code running on the workstations.

## 1 Introduction

We are working on pattern recognition problems using neural networks with a large number of parameters. Because of the large computational requirements of our area of research, we set out to design an integrated circuit that would serve as a good building block for our systems. Initially we considered designing extremely specialized chips, as this would maximize performance for a particular algorithm. However, the algorithms we use undergo considerable change as our research progresses. Still, we needed to provide some specialization if our design was to offer significant improvement over commercial workstation systems. Competing with workstations is

a challenge to anyone designing custom programmable processors, but as will be shown in this paper, one can still provide a performance advantage by focusing on one general class of computation.

Our solution was to design a vector microprocessor, T0, optimized for fixed-point computations, and to package this as an inexpensive workstation accelerator board. In this manner, we gain a considerable performance/cost advantage for neural network and other signal processing algorithms, while leveraging the commercial workstation environment for software development and I/O services.

In this paper, we focus on the neural network applications of the SPERT-II system. We are also investigating other applications in the areas of human-machine interface and multimedia processing, as we believe vector microprocessors show promise in providing the flexible, cost-effective, high-performance computing required.

Section 2 discusses the design of the hardware, followed in Section 3 by a discussion of the software environment we are developing and a discussion of related systems in Section 4. In Section 5 we discuss how we map a backpropagation training task to the system and in Section 6 we compare the resulting performance with two commercial workstation systems.

## 2    SPERT-II System

SPERT-II is a double slot SBus card for use in Sun compatible workstations and is shown in Figure 1. The board contains a T0 vector microprocessor and its memory, a Xilinx FPGA device for interfacing with the host, and various system support devices.

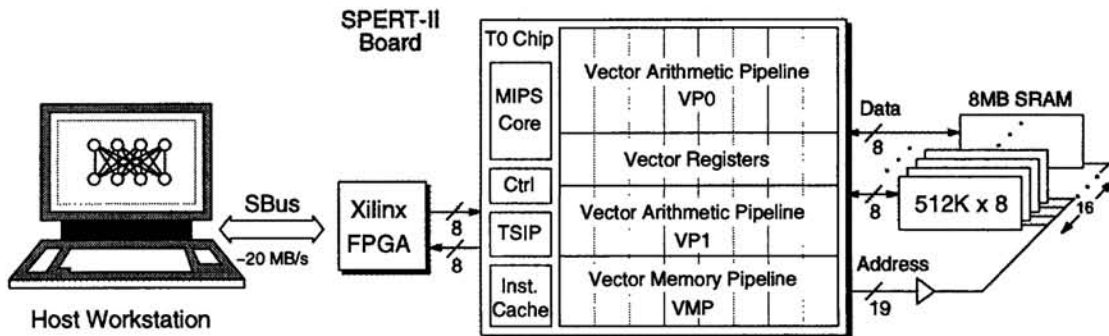

Figure 1: SPERT-II System Organization

### 2.1    The T0 vector microprocessor

Development of the T0 vector microprocessor follows our earlier work on the original SPERT VLIW/SIMD neuro-microprocessor (Wawrzynek, 1993). The most significant change we have made to the architecture is to move to a vector instruction set architecture (ISA), based on the industry standard MIPS RISC scalar ISA (Kane, 1992) extended with vector coprocessor instructions. The resulting ISA, which we call Torrent, offers important advantages over our previous design. We gain access to existing software tools for the MIPS architecture, including optimizing C compilers, assemblers, linkers, and debuggers. VLIW machines expose details of the hardware implementation at the instruction set level, and so must change instruction sets

when scaling to higher degrees of on-chip parallelism. In contrast, vector ISAs provide a simple abstraction of regular data parallelism that enables different hardware implementations to make different trade-offs between cost and performance while remaining software compatible. Compared with the VLIW/SIMD design, the vector ISA reduces requirements on instruction cache space and fetch bandwidth. It also makes it easier to write optimized library routines in assembly language, and these library routines will still run well on future devices with greater on-chip parallelism.

In the design of the T0 vector microprocessor, the main technique we employ to improve cost-performance over a commercial general purpose processor is to integrate multiple fixed-point datapaths with a high-bandwidth memory system. Fast digital arithmetic units, multipliers in particular, require chip area proportional to the *square* of the number of operand bits. In modern microprocessors and digital signal processors a single floating-point unit takes up a significant portion of the chip area. High-precision arithmetic units also requires high memory bandwidth to move large operands. However, for a wide class of problems, full-precision floating-point, or even high-precision fixed-point arithmetic, is not needed. Studies by ourselves and others have shown that for error back-propagation training of neural networks, 16-bit weights and 8-bit activation values provide similar training performance to IEEE single-precision floating-point (Asanović, 1991).

However, fast fixed-point multiply-adds alone are not sufficient to increase performance on a wide range of problems. Other components of a complete application may dominate total compute time if only multiply-add operations are accelerated. Our processor integrates a fast general-purpose RISC core, and includes general purpose operations in its vector instruction set to obtain a balanced design.

The T0 processor is a complete single chip implementation of the Torrent architecture. It was fabricated in Hewlett-Packard's CMOS26B process using $1.0\,\mu m$ scalable CMOS design rules and two layers of metal. The die measures 16.75mm × 16.75mm, and contains 730,701 transistors. T0 runs at an internal clock rate of 40MHz.

The main components of T0 are the MIPS-II compatible RISC CPU with an on-chip instruction cache, a vector unit coprocessor, a 128-bit wide external memory interface, and an 8-bit wide serial host interface (TSIP) and control unit. The external memory interface supports up to 4 GB of memory over a 128-bit wide data bus. The current SPERT-II board uses 16, 4 Mb SRAM parts to provide 8 MB of main memory.

At the core of the T0 processor is a MIPS-II compatible 32-bit integer RISC processor with a 1 KB instruction cache. The system coprocessor provides a 32-bit counter/timer and registers for host synchronization and exception handling.

The vector unit contains a vector register file with 16 vector registers, each holding 32 elements of 32 bits each, and three vector functional units, VP0, VP1, and VMP. VP0 and VP1 are vector arithmetic functional units. With the exception of multiplies, that must execute in VP0, either pipeline can execute any arithmetic operation. The multipliers perform 16-bit × 16-bit multiplies producing 32-bit results. All other arithmetic, logical and shift functions operate on 32 bits. VMP is the vector memory unit, and it handles all vector load/store operations, scalar load/store operations, and the vector insert/extract operations.

All three vector functional units are composed of 8 parallel pipelines, and so can each produce up to 8 results per cycle. The T0 memory interface has a single memory address port, therefore non-unit stride and indexed memory operations are limited to a rate of one element per cycle.

The elements of a vector register are striped across all 8 pipelines. With the maximum vector length of 32, a vector functional unit can accept a new instruction every 4 cycles. T0 can saturate all three vector functional units by issuing one instruction per cycle to each, leaving a single issue slot every 4 cycles for the scalar unit. In this manner, T0 can sustain up to 24 operations per cycle. Several important library routines, such as matrix-vector and matrix-matrix multiplies, have been written which achieve this level of performance. All vector pipeline hazards are fully interlocked in hardware, and so instruction scheduling is only required to improve performance, not to ensure correctness.

## 3  SPERT-II Software Environment

The primary design goal for the SPERT-II software environment was that it should appear as similar as possible to a conventional workstation environment. This should ease the task of porting existing workstation applications, as well as provide a comfortable environment for developing new code.

The Torrent instruction set architecture is based on the MIPS-II instruction set, with extra coprocessor instructions added to access the vector unit functionality. This compatibility allows us to base our software environment on the GNU tools which already include support for MIPS based machines. We have ported the **gcc** C/C++ compiler, modified the **gdb** symbolic debugger to debug T0 programs remotely from the host, enhanced the **gas** assembler to understand the new vector instructions and to schedule code to avoid interlocks, and we also employ the GNU linker and other library management utilities.

Currently, the only access to the vector unit we provide is either through library routines or directly via the scheduling assembler. We have developed an extensive set of optimized vector library routines including fixed-point matrix and vector operations, function approximation through linear interpolation, and IEEE single precision floating-point emulation. The majority of the routines are written in Torrent assembler, although a parallel set of functions have been written in ANSI C to allow program development and execution on workstations. Finally, there is a standard C library containing the usual utility, I/O and scalar math routines.

After compilation and linking, a T0 executable is run on the SPERT-II board by invoking a "server" program on the host. The server loads a small operating system "kernel" into T0 memory followed by the T0 executable. While the T0 application runs, the server services I/O requests on behalf of the T0 process.

## 4  Related Systems

Several programmable digital neurocomputers have been constructed, most notably systems based on the CNAPS chip from Adaptive Solutions (Hammerstrom, 1990) and the SYNAPSE-1, based on the MA-16 chip from Siemens (Ramacher, 1991).

The Adaptive Solutions CNAPS-1064 chip contains a SIMD array with 64 16-bit processing elements (PEs) per chip. Systems require an external microcode sequencer. The PEs have 16-bit datapaths with a single 32-bit accumulator, and are less flexible than the T0 datapaths. This chip provides on-chip memory for 128K 16-bit weights, distributed among the individual PEs. Off-chip memory bandwidth is limited by an 8-bit port. In contrast, T0 integrates an on-chip CPU that acts as controller, and provides fast access to a external memory equally accessible by all datapaths thereby increasing the range of applications that can be run efficiently.

Like SPERT-II, the SYNAPSE-1 leverages commercial memory parts. It features an array of MA-16 chips connected to interleaved DRAM memory banks. The MA-16 chips require extensive external circuitry, including 68040 CPUs with attached arithmetic pipelines, to execute computations not supported by the MA-16 itself. The SYNAPSE-1 system is a complex and expensive multi-board design, containing several different control streams that must be carefully orchestrated to run an application. However, for some applications the MA-16 could potentially provide greater throughput than T0 as the former's more specialized architecture permits more multiply–add units on each chip.

## 5   Mapping Backpropagation to T0

One artificial neural network (ANN) training task that we have done is taken from a speaker-independent continuous speech recognition system. The ANN is a simple feed-forward multi-layer perceptron (MLP) with three layers. Typical MLPs have between 100–400 input units. The input layer is fully connected to a hidden layer of 100–4000 hidden units. The hidden layer is fully connected to an output layer that contains one output per phoneme, typically 56–61. The hidden units incorporate a standard sigmoid activation function. The output units compute a "soft-max" activation function. All training is "on-line", with the weight matrices updated after each pattern presentation.

All of the compute-intensive sections can be readily vectorized on T0.

Three operations are performed on the weight matrices: forward propagation, error back-propagation, and weight update. These operations are available as three standard linear algebra routines in the T0 library: vector-matrix multiply, matrix-vector multiply, and scaled outer-product accumulation, respectively.

T0 can sustain one multiply-add per cycle in each of the 8 datapath slices, and can support this with one 16-bit memory access per cycle to each datapath slice provided that vector accesses have unit stride. The loops for the matrix operations are rearranged to perform only unit-stride memory accesses, and memory bandwidth requirements are further reduced by tiling matrix accesses and reusing operands from the vector registers whenever possible.

There are a number of other operations required while handling input and output vectors and activation values. While these require only $O(n)$ computation versus the $O(n^2)$ requirements of the matrix operations, they would present a significant overhead on smaller networks if not vectorized.

The sigmoid activation function is implemented using a library piecewise-linear function approximation routine. The function approximation routine makes use of the vector indexed load operations to perform the table lookups. Although T0 can only execute vector indexed operations at the rate of one element transfer per cycle, the table lookup routine can simultaneously perform all the arithmetic operations for index calculation and linear interpolation in the vector arithmetic units, achieving a rate of one 16-bit sigmoid result every 2 cycles. Similarly, a table based vector **logadd** routine is used to implement the soft-max function, also producing one result every 2 cycles.

To simplify software porting, the MLP code uses standard IEEE single-precision floating-point for input and output values. Vector library routines convert formats to the internal fixed-point representation. These conversion routines operate at the rate of up to 1 conversion every 2 cycles.

## 6    Performance Evaluation

We chose two commercial RISC workstations against which to compare the performance of the SPERT-II system. The first is a SPARCstation-20/61 containing a single 60 MHz SuperSPARC+ processor with a peak performance of 60 MFLOPS, 1 MB of second level cache, and 128 MB of DRAM main memory. The SPARCstation-20/61 is representative of a current mid-range workstation. The second is an IBM RS/6000-590, containing the RIOS-2 chipset running at 72 MHz with a peak performance of 266 MFLOPS, 256 KB of primary cache, and 768 MB of DRAM main memory. The RS/6000 is representative of a current high-end workstation.

The workstation version of the code performs all input and output and all computation using IEEE single precision floating-point arithmetic. The matrix and vector operations within the backprop algorithm have been extensively hand optimized, using manual loop unrolling together with register and cache blocking.

The SPERT-II numbers are obtained for a single T0 processor running at 40 MHz with 8 MB of SRAM main memory. The SPERT-II version of the application maintains the same interface, with input and output in IEEE single precision floating-point format, but performs all MLP computation using saturating fixed-point arithmetic with 16-bit weights, 16-bit activation values, and 32-bit intermediate results. The SPERT-II timings below include the time for conversion between floating-point and fixed-point for input and output.

Figure 2 shows the performance of the three systems for a set of three-layer networks on both backpropagation training and forward propagation. For ease of presentation we use networks with the same number of units per layer. Table 1 presents performance results for two speech network architectures. The general trend we observe in these evaluations is that for small networks the three hardware systems exhibit similar performance, while for larger network sizes the SPERT-II system demonstrates a significant performance advantage. For large networks the SPERT-II system demonstrates roughly 20–30 times the performance of a SPARC20 workstation and 4–6 times the performance of the IBM RS/6000-590 workstation.

### Acknowledgements

Thanks to Jerry Feldman for his contribution to the design of the SPERT-II system, Bertrand Irrisou for his work on the T0 chip, John Hauser for Torrent libraries, and John Lazzaro for his advice on chip and system building. Primary support for this work was from the ONR, URI Grant N00014-92-J-1617 and ARPA contract number N0001493-C0249. Additional support was provided by the NSF and ICSI.

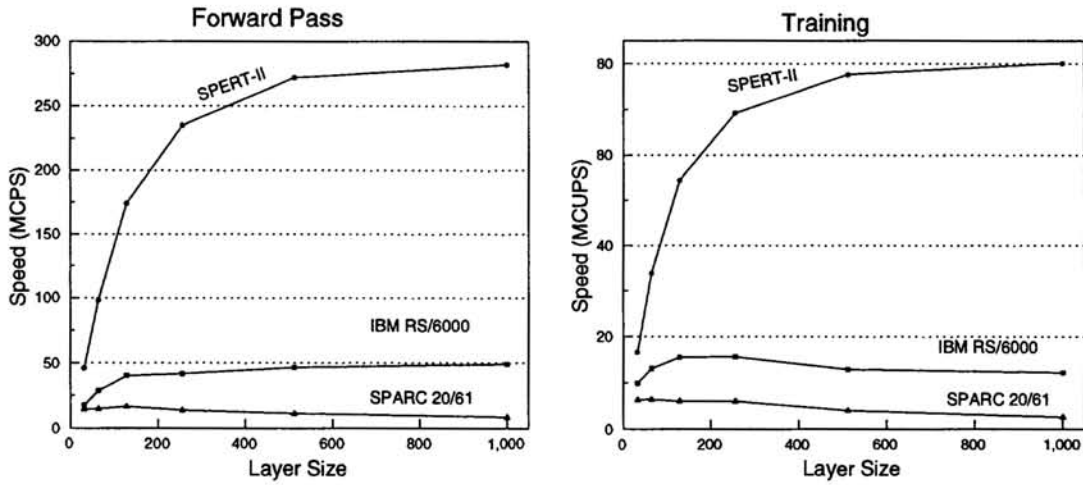

Figure 2: Performance Evaluation Results (all layers the same size).

Table 1: Performance Evaluation for Selected Net Sizes.

| net type | net size (in × hidden × out) | SPERT-II | SPARC20 | IBM RS/6000-590 |
|---|---|---|---|---|
| Forward Pass (MCPS) | | | | |
| small speech net | 153 × 200 × 56 | 181 | 17.6 | 43.0 |
| large speech net | 342 × 4000 × 61 | 276 | 11.3 | 45.1 |
| Training (MCUPS) | | | | |
| small speech net | 153 × 200 × 56 | 55.8 | 7.00 | 16.7 |
| large speech net | 342 × 4000 × 61 | 78.7 | 4.18 | 17.2 |

## References

Krste Asanović and Nelson Morgan. Experimental Determination of Precision Requirements for Back-Propagation Training of Artificial Neural Networks. In *Proc. 2nd Intl. Conf. on Microelectronics for Neural Networks*, Munich, Oct. 1991.

D. Hammerstrom. A VLSI architecture for High-Performance, Low-Cost, On-Chip Learning. In *Proc. Intl. Joint Conf. on Neural Networks*, pages II–537–543, 1990.

G. Kane, and Heinrich, J. *MIPS RISC Architecture*. Prentice Hall, 1992.

U. Ramacher, J. Beichter, W. Raab, J. Anlauf, N. Bruls, M. Hachmann, and M. Wesseling. Design of a 1st Generation Neurocomputer. In *VLSI Design of Neural Networks*. Kluwer Academic, 1991.

J. Wawrzynek, K. Asanović, and N. Morgan. The Design of a Neuro-Microprocessor. *IEEE Journal on Neural Networks*, 4(3), 1993.
